# Sample complexity of testing the manifold hypothesis

**Hariharan Narayanan**[*]
Laboratory for Information and Decision Systems
EECS, MIT
Cambridge, MA 02139
har@mit.edu
Sanjoy Mitter
Laboratory for Information and Decision Systems
EECS, MIT
Cambridge, MA 02139
mitter@mit.edu

## Abstract

The hypothesis that high dimensional data tends to lie in the vicinity of a low dimensional manifold is the basis of a collection of methodologies termed Manifold Learning. In this paper, we study statistical aspects of the question of fitting a manifold with a nearly optimal least squared error. Given upper bounds on the dimension, volume, and curvature, we show that Empirical Risk Minimization can produce a nearly optimal manifold using a number of random samples that is *independent* of the ambient dimension of the space in which data lie. We obtain an upper bound on the required number of samples that depends polynomially on the curvature, exponentially on the intrinsic dimension, and linearly on the intrinsic volume. For constant error, we prove a matching minimax lower bound on the sample complexity that shows that this dependence on intrinsic dimension, volume and curvature is unavoidable. Whether the known lower bound of $O(\frac{k}{\epsilon^2} + \frac{\log\frac{1}{\delta}}{\epsilon^2})$ for the sample complexity of Empirical Risk minimization on $k-$means applied to data in a unit ball of arbitrary dimension is tight, has been an open question since 1997 [3]. Here $\epsilon$ is the desired bound on the error and $\delta$ is a bound on the probability of failure. We improve the best currently known upper bound [14] of $O(\frac{k^2}{\epsilon^2} + \frac{\log\frac{1}{\delta}}{\epsilon^2})$ to $O\left(\frac{k}{\epsilon^2}\left(\min\left(k, \frac{\log^4\frac{k}{\epsilon}}{\epsilon^2}\right)\right) + \frac{\log\frac{1}{\delta}}{\epsilon^2}\right)$. Based on these results, we devise a simple algorithm for $k-$means and another that uses a family of convex programs to fit a piecewise linear curve of a specified length to high dimensional data, where the sample complexity is independent of the ambient dimension.

## 1 Introduction

We are increasingly confronted with very high dimensional data in speech signals, images, gene-expression data, and other sources. Manifold Learning can be loosely defined to be a collection of methodologies that are motivated by the belief that this hypothesis (henceforth called the manifold hypothesis) is true. It includes a number of extensively used algorithms such as Locally Linear Embedding [17], ISOMAP [19], Laplacian Eigenmaps [4] and Hessian Eigenmaps [8]. The sample complexity of classification is known to be independent of the ambient dimension [15] under the manifold hypothesis, (assuming the decision boundary is a manifold as well,) thus obviating the curse of dimensionality. A recent empirical study [6] of a large number of $3 \times 3$ images, represented as points in $\mathbb{R}^9$ revealed that they approximately lie on a two dimensional manifold known as the

---

[*]Research supported by grant CCF-0836720

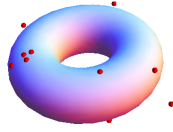

Figure 1: Fitting a torus to data.

Klein bottle. On the other hand, knowledge that the manifold hypothesis is false with regard to certain data would give us reason to exercise caution in applying algorithms from manifold learning and would provide an incentive for further study.

It is thus of considerable interest to know whether given data lie in the vicinity of a low dimensional manifold. Our primary technical results are the following.

1. We obtain uniform bounds relating the empirical squared loss and the true squared loss over a class $\mathcal{F}$ consisting of manifolds whose dimensions, volumes and curvatures are bounded in Theorems 1 and 2. These bounds imply upper bounds on the sample complexity of Empirical Risk Minimization (ERM) that are *independent* of the ambient dimension, exponential in the intrinsic dimension, polynomial in the curvature and almost linear in the volume.

2. We obtain a minimax lower bound on the sample complexity of *any* rule for learning a manifold from $\mathcal{F}$ in Theorem 6 showing that for a fixed error, the the dependence of the sample complexity on intrinsic dimension, curvature and volume must be at least exponential, polynomial, and linear, respectively.

3. We improve the best currently known upper bound [14] on the sample complexity of Empirical Risk minimization on $k-$means applied to data in a unit ball of arbitrary dimension from $O(\frac{k^2}{\epsilon^2} + \frac{\log\frac{1}{\delta}}{\epsilon^2})$ to $O\left(\frac{k}{\epsilon^2}\left(\min\left(k, \frac{\log^4\frac{k}{\epsilon}}{\epsilon^2}\right)\right) + \frac{\log\frac{1}{\delta}}{\epsilon^2}\right)$. Whether the known lower bound of $O(\frac{k}{\epsilon^2} + \frac{\log\frac{1}{\delta}}{\epsilon^2})$ is tight, has been an open question since 1997 [3]. Here $\epsilon$ is the desired bound on the error and $\delta$ is a bound on the probability of failure.

One technical contribution of this paper is the use of dimensionality reduction via random projections in the proof of Theorem 5 to bound the Fat-Shattering dimension of a function class, elements of which roughly correspond to the squared distance to a low dimensional manifold. The application of the probabilistic method involves a projection onto a low dimensional random subspace. This is then followed by arguments of a combinatorial nature involving the VC dimension of halfspaces, and the Sauer-Shelah Lemma applied with respect to the low dimensional subspace. While random projections have frequently been used in machine learning algorithms, for example in [2, 7], to our knowledge, they have not been used as a tool to bound the complexity of a function class. We illustrate the algorithmic utility of our uniform bound by devising an algorithm for $k-$means and a convex programming algorithm for fitting a piecewise linear curve of bounded length. For a fixed error threshold and length, the dependence on the ambient dimension is linear, which is optimal since this is the complexity of reading the input.

## 2    Connections and Related work

In the context of curves, [10] proposed "Principal Curves", where it was suggested that a natural curve that may be fit to a probability distribution is one where every point on the curve is the center of mass of all those points to which it is the nearest point. A different definition of a principal curve was proposed by [12], where they attempted to find piecewise linear curves of bounded length which minimize the expected squared distance to a random point from a distribution. This paper studies the decay of the error rate as the number of samples tends to infinity, but does not analyze the dependence of the error rate on the ambient dimension and the bound on the length. We address this in a more general setup in Theorem 4, and obtain sample complexity bounds that are independent of

the ambient dimension, and depend linearly on the bound on the length. There is a significant amount of recent research aimed at understanding topological aspects of data, such its homology [20, 16]. It has been an open question since 1997 [3], whether the known lower bound of $O(\frac{k}{\epsilon^2} + \frac{\log\frac{1}{\delta}}{\epsilon^2})$ for the sample complexity of Empirical Risk minimization on $k-$means applied to data in a unit ball of arbitrary dimension is tight. Here $\epsilon$ is the desired bound on the error and $\delta$ is a bound on the probability of failure. The best currently known upper bound is $O(\frac{k^2}{\epsilon^2} + \frac{\log\frac{1}{\delta}}{\epsilon^2})$ and is based on Rademacher complexities. We improve this bound to $O\left(\frac{k}{\epsilon^2}\left(\min\left(k, \frac{\log^4\frac{k}{\epsilon}}{\epsilon^2}\right)\right) + \frac{\log\frac{1}{\delta}}{\epsilon^2}\right)$, using an argument that bounds the Fat-Shattering dimension of the appropriate function class using random projections and the Sauer-Shelah Lemma. Generalizations of principal curves to parameterized principal manifolds in certain regularized settings have been studied in [18]. There, the sample complexity was related to the decay of eigenvalues of a Mercer kernel associated with the regularizer. When the manifold to be fit is a set of $k$ points ($k-$means), we obtain a bound on the sample complexity $s$ that is independent of $m$ and depends at most linearly on $k$, which also leads to an approximation algorithm with additive error, based on sub-sampling. If one allows a multiplicative error of $4$ in addition to an additive error of $\epsilon$, a statement of this nature has been proven by Ben-David (Theorem 7, [5]).

## 3 Upper bounds on the sample complexity of Empirical Risk Minimization

In the remainder of the paper, $C$ will always denote a universal constant which may differ across the paper. For any submanifold $\mathcal{M}$ contained in, and probability distribution $\mathcal{P}$ supported on the unit ball $B$ in $\mathbb{R}^m$, let $\mathcal{L}(\mathcal{M}, \mathcal{P}) := \int \mathbf{d}(\mathcal{M}, x)^2 d\mathcal{P}(x)$. Given a set of i.i.d points $x = \{x_1, \ldots, x_s\}$ from $\mathcal{P}$, a tolerance $\epsilon$ and a class of manifolds $\mathcal{F}$, Empirical Risk Minimization (ERM) outputs a manifold in $\mathcal{M}_{erm}(x) \in \mathcal{F}$ such that $\sum_{i=1}^{s} \mathbf{d}(x_i, \mathcal{M}_{erm})^2 \leq \epsilon/2 + \inf_{\mathcal{N} \in \mathcal{F}} \mathbf{d}(x_i, \mathcal{N})^2$. Given error parameters $\epsilon, \delta$, and a rule $\mathcal{A}$ that outputs a manifold in $\mathcal{F}$ when provided with a set of samples, we define the sample complexity $s = s(\epsilon, \delta, \mathcal{A})$ to be the least number such that for any probability distribution $\mathcal{P}$ in the unit ball, if the result of $\mathcal{A}$ applied to a set of at least $s$ i.i.d random samples from $\mathcal{P}$ is $\mathcal{N}$, then $\mathbb{P}\left[\mathcal{L}(\mathcal{N}, \mathcal{P}) < \inf_{\mathcal{M} \in \mathcal{F}} \mathcal{L}(\mathcal{M}, \mathcal{P}) + \epsilon\right] > 1 - \delta$.

### 3.1 Bounded intrinsic curvature

Let $\mathcal{M}$ be a Riemannian manifold and let $p \in \mathcal{M}$. Let $\zeta$ be a geodesic starting at $p$.

**Definition 1.** *The first point on $\zeta$ where $\zeta$ ceases to minimize distance is called the cut point of $p$ along $\mathcal{M}$. The cut locus of $p$ is the set of cut points of $\mathcal{M}$. The injectivity radius is the minimum taken over all points of the distance between the point and its cut locus. $\mathcal{M}$ is complete if it is complete as a metric space.*

Let $\mathcal{G}_i = \mathcal{G}_i(d, V, \lambda, \iota)$ be the family of all isometrically embedded, complete Riemannian submanifolds of $B$ having dimension less or equal to $d$, induced $d-$dimensional volume less or equal to $V$, sectional curvature less or equal to $\lambda$ and injectivity radius greater or equal to $\iota$. Let $U_{int}(\frac{1}{\epsilon}, d, V, \lambda, \iota) := V\left(C\left(\frac{d}{\min(\epsilon, \lambda^{-1/2})}\right)\right)^d$, which for brevity, we denote $U_{int}$.

**Theorem 1.** *Let $\epsilon$ and $\delta$ be error parameters. If*

$$s \geq C\left(\min\left(\left(\frac{1}{\epsilon^2}\right)\log^4\left(\frac{U_{int}}{\epsilon}\right), U_{int}\right)\frac{U_{int}}{\epsilon^2} + \frac{1}{\epsilon^2}\log\frac{1}{\delta}\right),$$

*and $x = \{x_1, \ldots, x_s\}$ is a set of i.i.d points from $\mathcal{P}$ then,*

$$\mathbb{P}\left[\mathcal{L}(\mathcal{M}_{erm}(x), \mathcal{P}) - \inf_{\mathcal{M} \in \mathcal{G}_i} \mathcal{L}(\mathcal{M}, \mathcal{P}) < \epsilon\right] > 1 - \delta.$$

The proof of this theorem is deferred to Section 4.

### 3.2 Bounded extrinsic curvature

We will consider submanifolds of $B$ that have the property that around each of them, for any radius $r < \tau$, the boundary of the set of all points within a distance $r$ is smooth. This class of submanifolds

has appeared in the context of manifold learning [16, 15]. The condition number is defined as follows.

**Definition 2** (Condition Number). *Let $\mathcal{M}$ be a smooth $d-$dimensional submanifold of $\mathbb{R}^m$. We define the condition number $c(\mathcal{M})$ to be $\frac{1}{\tau}$, where $\tau$ is the largest number to have the property that for any $r < \tau$ no two normals of length $r$ that are incident on $\mathcal{M}$ have a common point unless it is on $\mathcal{M}$.*

Let $\mathcal{G}_e = \mathcal{G}_e(d, V, \tau)$ be the family of Riemannian submanifolds of $B$ having dimension $\leq d$, volume $\leq V$ and condition number $\leq \frac{1}{\tau}$. Let $\epsilon$ and $\delta$ be error parameters. Let $U_{ext}(\frac{1}{\epsilon}, d, \tau) := V \left( C \left( \frac{d}{\min(\epsilon, \tau)} \right) \right)^d$, which for brevity, we denote by $U_{ext}$.

**Theorem 2.** *If*

$$s \geq C \left( \min \left( \left( \frac{1}{\epsilon^2} \right) \log^4 \left( \frac{U_{ext}}{\epsilon} \right), U_{ext} \right) \frac{U_{ext}}{\epsilon^2} + \frac{1}{\epsilon^2} \log \frac{1}{\delta} \right),$$

*and $x = \{x_1, \ldots, x_s\}$ is a set of i.i.d points from $\mathcal{P}$ then,*

$$\mathbb{P} \left[ \mathcal{L}(\mathcal{M}_{erm}(x), \mathcal{P}) - \inf_{\mathcal{M}} \mathcal{L}(\mathcal{M}, \mathcal{P}) < \epsilon \right] > 1 - \delta. \tag{1}$$

## 4 Relating bounded curvature to covering number

In this subsection, we note that that bounds on the dimension, volume, sectional curvature and injectivity radius suffice to ensure that they can be covered by relatively few $\epsilon-$balls. Let $V_p^{\mathcal{M}}$ be the volume of a ball of radius $\mathcal{M}$ centered around a point $p$. See ([9], page 51) for a proof of the following theorem.

**Theorem 3** (Bishop-Günther Inequality). *Let $\mathcal{M}$ be a complete Riemannian manifold and assume that $r$ is not greater than the injectivity radius $\iota$. Let $K^{\mathcal{M}}$ denote the sectional curvature of $\mathcal{M}$ and let $\lambda > 0$ be a constant. Then, $K^{\mathcal{M}} \leq \lambda$ implies $V_p^{\mathcal{M}}(r) \geq \frac{2\pi^{\frac{n}{2}}}{\Gamma\left(\frac{n}{2}\right)} \int_0^r \left( \frac{\sin(t\sqrt{\lambda})}{\sqrt{\lambda}} \right)^{n-1} dt.$*

Thus, if $\epsilon < \min(\iota, \frac{\pi \lambda^{-\frac{1}{2}}}{2})$, then, $V_p^{\mathcal{M}}(\epsilon) > \left( \frac{\epsilon}{Cd} \right)^d$.

*Proof of Theorem 1.* As a consequence of Theorem 3, we obtain an upper bound of $V \left( \frac{Cd}{\epsilon} \right)^d$ on the number of disjoint sets of the form $\mathcal{M} \cap B_{\epsilon/32}(p)$ that can be packed in $\mathcal{M}$. If $\{\mathcal{M} \cap B_{\epsilon/32}(p_1), \ldots, \mathcal{M} \cap B_{\epsilon/32}(p_k)\}$ is a maximal family of disjoint sets of the form $\mathcal{M} \cap B_{\epsilon/32}(p)$, then there is no point $p \in \mathcal{M}$ such that $\min_i \|p - p_i\| > \epsilon/16$. Therefore, $\mathcal{M}$ is contained in the union of balls, $\bigcup_i B_{\epsilon/16}(p_i)$. Therefore, we may apply Theorem 4 with $U\left(\frac{1}{\epsilon}\right) \leq V \left( \frac{Cd}{\min(\epsilon, \lambda^{-\frac{1}{2}}, \iota)} \right)^d.$ □

The proof of Theorem 2 is along the lines of that of Theorem 1, so it has been deferred to the journal version.

## 5 Class of manifolds with a bounded covering number

In this section, we show that uniform bounds relating the empirical squares loss and the expected squared loss can be obtained for a class of manifolds whose covering number at a different scale $\epsilon$ has a specified upper bound. Let $U : \mathbb{R}^+ \to \mathbb{Z}^+$ be any integer valued function. Let $\mathcal{G}$ be any family of subsets of $B$ such that for all $r > 0$ every element $\mathcal{M} \in \mathcal{G}$ can be covered using open Euclidean balls of radius $r$ centered around $U(\frac{1}{r})$ points; let this set be $\Lambda_{\mathcal{M}}(r)$. Note that if the subsets consist of $k-$tuples of points, $U(1/r)$ can be taken to be the constant function equal to $k$ and we recover the $k-$means question. A priori, it is unclear if

$$\sup_{\mathcal{M} \in \mathcal{G}} \left| \frac{\sum_{i=1}^s \mathbf{d}(x_i, \mathcal{M})^2}{s} - \mathbb{E}_{\mathcal{P}} \mathbf{d}(x, \mathcal{M})^2 \right|, \tag{2}$$

is a random variable, since the supremum of a set of random variables is not always a random variable (although if the set is countable this is true). However (2) is equal to

$$\lim_{n \to \infty} \sup_{\mathcal{M} \in \mathcal{G}} \left| \frac{\sum_{i=1}^s \mathbf{d}(x_i, \Lambda_{\mathcal{M}}(1/n))^2}{s} - \mathbb{E}_{\mathcal{P}} \mathbf{d}(x, \Lambda_{\mathcal{M}}(1/n))^2 \right|, \tag{3}$$

and for each $n$, the supremum in the limits is over a set parameterized by $U(n)$ points, which without loss of generality we may take to be countable (due to the density and countability of rational points). Thus, for a fixed $n$, the quantity in the limits is a random variable. Since the limit as $n \to \infty$ of a sequence of bounded random variables is a random variable as well, (2) is a random variable too.

**Theorem 4.** *Let $\epsilon$ and $\delta$ be error parameters. If*

$$s \geq C \left( \frac{U(16/\epsilon)}{\epsilon^2} \min \left( U(16/\epsilon), \left( \frac{1}{\epsilon^2} \right) \log^4 \left( \frac{U(16/\epsilon)}{\epsilon} \right) \right) + \frac{1}{\epsilon^2} \log \frac{1}{\delta} \right),$$

*Then,*

$$\mathbb{P} \left[ \sup_{\mathcal{M} \in \mathcal{G}} \left| \frac{\sum_{i=1}^s \mathbf{d}(x_i, \mathcal{M})^2}{s} - \mathbb{E}_{\mathcal{P}} \mathbf{d}(x, \mathcal{M})^2 \right| < \frac{\epsilon}{2} \right] > 1 - \delta. \tag{4}$$

*Proof.* For every $g \in \mathcal{G}$, let $\mathbf{c}(g, \epsilon) = \{c_1, \ldots, c_k\}$ be a set of $k := U(16/\epsilon)$ points in $g \subseteq B$, such that $g$ is covered by the union of balls of radius $\epsilon/16$ centered at these points. Thus, for any point $x \in B$,

$$\mathbf{d}^2(x, g) \quad \leq \quad \left( \frac{\epsilon}{16} + \mathbf{d}(x, \mathbf{c}(g, \epsilon)) \right)^2 \tag{5}$$

$$\leq \quad \frac{\epsilon^2}{256} + \frac{\epsilon \min_i \|x - c_i\|}{8} + \mathbf{d}(x, \mathbf{c}(g, \epsilon))^2. \tag{6}$$

Since $\min_i \|x - c_i\|$ is less or equal to 2, the last expression is less than $\frac{\epsilon}{2} + \mathbf{d}(x, \mathbf{c}(g, \epsilon))^2$. Our proof uses the "kernel trick" in conjunction with Theorem 5. Let $\Phi : (x_1, \ldots, x_m)^T \mapsto 2^{-1/2}(x_1, \ldots, x_m, 1)^T$ map a point $x \in \mathbb{R}^m$ to one in $\mathbb{R}^{m+1}$. For each $i$, let $c_i := (c_{i1}, \ldots, c_{im})^T$, and $\tilde{c}_i := 2^{-1/2}(-c_{i1}, \ldots, -c_{im}, \frac{\|c_i\|^2}{2})^T$. The factor of $2^{-1/2}$ is necessitated by the fact that we wish the image of a point in the unit ball to also belong to the unit ball. Given a collection of points $\mathbf{c} := \{c_1, \ldots, c_k\}$ and a point $x \in B$, let $f_{\mathbf{c}}(x) := \mathbf{d}(x, \mathbf{c}(g, \epsilon))^2$. Then,

$$f_{\mathbf{c}}(x) = \|x\|^2 + 4 \min(\Phi(x) \cdot \tilde{c}_1, \ldots, \Phi(x) \cdot \tilde{c}_k).$$

For any set of $s$ samples $x_1, \ldots, x_s$,

$$\sup_{f_{\mathbf{c}} \in \mathcal{G}} \left| \frac{\sum_{i=1}^s f_{\mathbf{c}}(x_i)}{s} - \mathbb{E}_{\mathcal{P}} f_{\mathbf{c}}(x) \right| \quad \leq \quad \left| \frac{\sum_{i=1}^s \|x_i\|^2}{s} - \mathbb{E}_{\mathcal{P}} \|x\|^2 \right| \tag{7}$$

$$+ \quad 4 \sup_{f_{\mathbf{c}} \in \mathcal{G}} \left| \frac{\sum_{i=1}^s \min_i \Phi(x_i) \cdot \tilde{c}_i}{s} - \mathbb{E}_{\mathcal{P}} \min_i \Phi(x) \cdot \tilde{c}_i \right|. \tag{8}$$

By Hoeffding's inequality,

$$\mathbb{P} \left[ \left| \frac{\sum_{i=1}^s \|x_i\|^2}{s} - \mathbb{E}_{\mathcal{P}} \|x\|^2 \right| > \frac{\epsilon}{4} \right] < 2e^{-\left(\frac{1}{8}\right)s\epsilon^2}, \tag{9}$$

which is less than $\frac{\delta}{2}$.

By Theorem 5, $\mathbb{P} \left[ \sup_{f_{\mathbf{c}} \in \mathcal{G}} \left| \frac{\sum_{i=1}^s \min_i \Phi(x_i) \cdot \tilde{c}_i}{s} - \mathbb{E}_{\mathcal{P}} \min_i \Phi(x) \cdot \tilde{c}_i \right| > \frac{\epsilon}{16} \right] < \frac{\delta}{2}.$

Therefore, $\mathbb{P} \left[ \sup_{f_{\mathbf{c}} \in \mathcal{G}} \left| \frac{\sum_{i=1}^s f_{\mathbf{c}}(x_i)}{s} - \mathbb{E}_{\mathcal{P}} f_{\mathbf{c}}(x) \right| \leq \frac{\epsilon}{2} \right] \geq 1 - \delta.$

$\square$

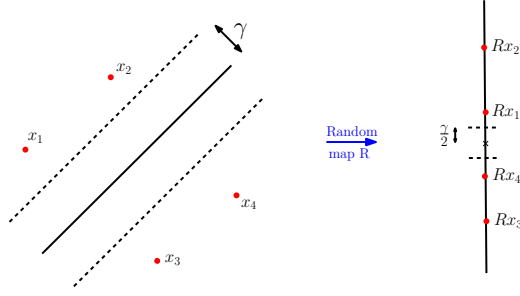

Figure 2: Random projections are likely to preserve linear separations.

# 6    Bounding the Fat-Shattering dimension using random projections

The core of the uniform bounds in Theorems 1 and 2 is the following uniform bound on the minimum of $k$ linear functions on a ball in $\mathbb{R}^m$.

**Theorem 5.** *Let $\mathcal{F}$ be the set of all functions $f$ from $B := \{x \in \mathbb{R}^m : \|x\| \leq 1\}$ to $\mathbb{R}$, such that for some $k$ vectors $v_1, \ldots, v_k \in B$,*

$$f(x) := \min_i (v_i \cdot x).$$

*Independent of $m$, if*

$$s \geq C \left( \frac{k}{\epsilon^2} \min \left( \frac{1}{\epsilon^2} \log^4 \left( \frac{k}{\epsilon} \right), k \right) + \frac{1}{\epsilon^2} \log \frac{1}{\delta} \right),$$

*then*

$$\mathbb{P} \left[ \sup_{F \in \mathcal{F}} \left| \frac{\sum_{i=1}^s F(x_i)}{s} - \mathbb{E}_{\mathcal{P}} F(x) \right| < \epsilon \right] > 1 - \delta. \tag{10}$$

It has been open since 1997 [3], whether the known lower bound of $C \left( \frac{k}{\epsilon^2} + \frac{1}{\epsilon^2} \log \frac{1}{\delta} \right)$ on the sample complexity $s$ is tight. Theorem 5 in [14], uses Rademacher complexities to obtain an upper bound of

$$C \left( \frac{k^2}{\epsilon^2} + \frac{1}{\epsilon^2} \log \frac{1}{\delta} \right). \tag{11}$$

(The scenarios in [3, 14] are that of $k-$means, but the argument in Theorem 4 reduces $k-$means to our setting.) Theorem 5 improves this to

$$C \left( \frac{k}{\epsilon^2} \min \left( \frac{1}{\epsilon^2} \log^4 \left( \frac{k}{\epsilon} \right), k \right) + \frac{1}{\epsilon^2} \log \frac{1}{\delta} \right) \tag{12}$$

by putting together (11) with a bound of

$$C \left( \frac{k}{\epsilon^4} \log^4 \left( \frac{k}{\epsilon} \right) + \frac{1}{\epsilon^2} \log \frac{1}{\delta} \right) \tag{13}$$

obtained using the Fat-Shattering dimension. Due to constraints on space, the details of the proof of Theorem 5 will appear in the journal version, but the essential ideas are summarized here.

Let $u := \mathrm{fat}_{\mathcal{F}}(\frac{\epsilon}{24})$ and $x_1, \ldots, x_u$ be a set of vectors that is $\gamma-$shattered by $\mathcal{F}$. We would like to use VC theory to bound $u$, but doing so directly leads to a linear dependence on the ambient dimension $m$. In order to circumvent this difficulty, for $g := C \frac{\log(u+k)}{\epsilon^2}$, we consider a $g-$dimensional random linear subspace and the image under an appropriately scaled orthogonal projection $R$ of the points $x_1, \ldots, x_u$ onto it. We show that the expected value of the $\frac{\gamma}{2}-$shatter coefficient of $\{Rx_1, \ldots, Rx_u\}$ is at least $2^{u-1}$ using the Johnson-Lindenstrauss Lemma [11] and the fact that $\{x_1, \ldots, x_u\}$ is $\gamma-$shattered. Using Vapnik-Chervonenkis theory and the Sauer-Shelah Lemma, we then show that $\frac{\gamma}{2}-$shatter coefficient cannot be more than $u^{k(g+2)}$. This implies that $2^{u-1} \leq u^{k(g+2)}$, allowing us to conclude that $\mathrm{fat}_{\mathcal{F}}(\frac{\epsilon}{24}) \leq \frac{Ck}{\epsilon^2} \log^2 \left( \frac{k}{\epsilon} \right)$. By a well-known theorem of [1], a bound of $\frac{Ck}{\epsilon^2} \log^2 \left( \frac{k}{\epsilon} \right)$ on $\mathrm{fat}_{\mathcal{F}}(\frac{\epsilon}{24})$ implies the bound in (13) on the sample complexity, which implies Theorem 5.

# 7 Minimax lower bounds on the sample complexity

Let $K$ be a universal constant whose value will be fixed throughout this section. In this section, we will state lower bounds on the number of samples needed for the minimax decision rule for learning from high dimensional data, with high probability, a manifold with a squared loss that is within $\epsilon$ of the optimal. We will construct a carefully chosen prior on the space of probability distributions and use an argument that can either be viewed as an application of the probabilistic method or of the fact that the Minimax risk is at least the risk of a Bayes optimal manifold computed with respect to this prior. Let $U$ be a $K^{2d}k-$dimensional vector space containing the origin, spanned by the basis $\{e_1, \ldots, e_{K^{2d}k}\}$ and $S$ be the surface of the ball of radius 1 in $\mathbb{R}^m$. We assume that $m$ be greater or equal to $K^{2d}k+d$. Let $W$ be the $d-$dimensional vector space spanned by $\{e_{K^{2d}k+1}, \ldots, e_{K^{2d}k+d}\}$. Let $S_1, \ldots, S_{K^{2d}k}$ denote spheres, such that for each $i$, $S_i := S \cap (\sqrt{1-\tau^2}e_i + W)$, where $x + W$ is the translation of $W$ by $x$. Note that each $S_i$ has radius $\tau$. Let $\ell = \binom{K^{2d}k}{K^d k}$ and $\{\mathcal{M}_1, \ldots, \mathcal{M}_\ell\}$ consist of all $K^d k-$element subsets of $\{S_1, \ldots, S_{K^{2d}k}\}$. Let $\omega_d$ be the volume of the unit ball in $\mathbb{R}^d$. The following theorem shows that no algorithm can produce a nearly optimal manifold with high probability unless it uses a number of samples that depends linearly on volume, exponentially on intrinsic dimension and polynomially on the curvature.

**Theorem 6.** *Let $\mathcal{F}$ be equal to either $\mathcal{G}_e(d, V, \tau)$ or $\mathcal{G}_i(d, V, \frac{1}{\tau^2}, \pi\tau)$. Let $k = \lfloor \frac{V}{d\omega_d(K^{\frac{5}{4}}\tau)^d} \rfloor$. Let $\mathcal{A}$ be an arbitrary algorithm that takes as input a set of data points $x = \{x_1, \ldots, x_k\}$ and outputs a manifold $\mathcal{M}_{\mathcal{A}}(x)$ in $\mathcal{F}$. If $\epsilon + 2\delta < \frac{1}{3}\left(\frac{1}{2\sqrt{2}} - \tau\right)^2$ then,*

$$\inf_{\mathcal{P}} \mathbb{P}\left[\mathcal{L}(\mathcal{M}_{\mathcal{A}}(x), \mathcal{P}) - \inf_{\mathcal{M}\in\mathcal{F}}\mathcal{L}(\mathcal{M}, \mathcal{P}) < \epsilon\right] < 1 - \delta,$$

*where $\mathcal{P}$ ranges over all distributions supported on $B$ and $x_1, \ldots, x_k$ are i.i.d draws from $\mathcal{P}$.*

*Proof.* Observe from Lemma **??** and Theorem 3 that $\mathcal{F}$ is a class of a manifolds such that each manifold in $\mathcal{F}$ is contained in the union of $K^{\frac{3d}{2}}k$ $m-$dimensional balls of radius $\tau$, and $\{\mathcal{M}_1, \ldots, \mathcal{M}_\ell\} \subseteq \mathcal{F}$. (The reason why we have $K^{\frac{3d}{2}}$ rather than $K^{\frac{5d}{4}}$ as in the statement of the theorem is that the parameters of $\mathcal{G}_i(d, V, \tau)$ are intrinsic, and to transfer to the extrinsic setting of the last sentence, one needs some leeway.) Let $\mathcal{P}_1, \ldots, \mathcal{P}_\ell$ be probability distributions that are uniform on $\{\mathcal{M}_1, \ldots, \mathcal{M}_\ell\}$ with respect to the induced Riemannian measure. Suppose $A$ is an algorithm that takes as input a set of data points $x = \{x_1, \ldots, x_t\}$ and outputs a manifold $\mathcal{M}_{\mathcal{A}}(x)$. Let $r$ be chosen uniformly at random from $\{1, \ldots, \ell\}$. Then,

$$\begin{aligned}
\inf_{\mathcal{P}} \mathbb{P}\left[\left|\mathcal{L}(\mathcal{M}_{\mathcal{A}}(x), \mathcal{P}) - \inf_{\mathcal{M}\in\mathcal{F}}\mathcal{L}(\mathcal{M}, \mathcal{P})\right| < \epsilon\right] &\leq \mathbb{E}_{\mathcal{P}_r}\mathbb{P}_x\left[\left|\mathcal{L}(\mathcal{M}_{\mathcal{A}}(x), \mathcal{P}_r) - \inf_{\mathcal{M}\in\mathcal{F}}\mathcal{L}(\mathcal{M}, \mathcal{P}_r)\right| < \epsilon\right] \\
&= \mathbb{E}_x\mathbb{P}_{\mathcal{P}_r}\left[\left|\mathcal{L}(\mathcal{M}_{\mathcal{A}}(x), \mathcal{P}_r) - \inf_{\mathcal{M}\in\mathcal{F}}\mathcal{L}(\mathcal{M}, \mathcal{P}_r)\right| < \epsilon\big|x\right] \\
&= \mathbb{E}_x\mathbb{P}_{\mathcal{P}_r}\left[\mathcal{L}(\mathcal{M}_{\mathcal{A}}(x), \mathcal{P}_r) < \epsilon\big|x\right].
\end{aligned}$$

We first prove a lower bound on $\inf_x \mathbb{E}_r\left[\mathcal{L}(\mathcal{M}_{\mathcal{A}}(x), \mathcal{P}_r)|x\right]$.

We see that

$$\mathbb{E}_r\left[\mathcal{L}(\mathcal{M}_{\mathcal{A}}(x), \mathcal{P}_r)\big|x\right] = \mathbb{E}_{r,x_{k+1}}\left[\mathbf{d}(\mathcal{M}_{\mathcal{A}}(x), x_{k+1})^2\big|x\right]. \tag{14}$$

Conditioned on $x$, the probability of the event (say $E_{dif}$) that $x_{k+1}$ does not belong to the same sphere as one of the $x_1, \ldots, x_k$ is at least $\frac{1}{2}$.

Conditioned on $E_{dif}$ and $x_1, \ldots, x_k$, the probability that $x_{k+1}$ lies on a given sphere $S_j$ is equal to 0 if one of $x_1, \ldots, x_k$ lies on $S_j$ and $\frac{1}{K^2 k - k'}$ otherwise, where $k' \leq k$ is the number of spheres in $\{S_i\}$ which contain at least one point among $x_1, \ldots, x_k$.

By construction, $A(x_1, \ldots, x_k)$ can be covered by $K^{\frac{3d}{2}}k$ balls of radius $\tau$; let their centers be $y_1, \ldots, y_{K^{\frac{3d}{2}}k}$.

However, it is easy to check that for any dimension $m$, the cardinality of the set $\mathcal{S}_y$ of all $S_i$ that have a nonempty intersection with the balls of radius $\frac{1}{2\sqrt{2}}$ centered around $y_1, \ldots, y_{K^{\frac{3d}{2}}k}$, is at most $K^{\frac{3d}{2}}k$. Therefore, $\mathbb{P}\left[\mathbf{d}(\mathcal{M}_\mathcal{A}(x), x_{k+1}) \geq \frac{1}{2\sqrt{2}} - \tau \big| x\right]$ is at least

$$\mathbb{P}\left[\mathbf{d}(\{y_1, \ldots, y_{K^{\frac{3d}{2}}k}\}, x_{k+1}) \geq \frac{1}{2\sqrt{2}} \big| x\right] \geq \mathbb{P}\left[E_{dif}\right] \mathbb{P}\left[x_{k+1} \notin \mathcal{S}_y | E_{dif}\right]$$

$$\geq \frac{1}{2} \frac{K^{2d}k - k' - K^{\frac{3d}{2}}k}{K^{2d}k - k'}$$

$$\geq \frac{1}{3}.$$

Therefore, $\mathbb{E}_{r, x_{k+1}}\left[\mathbf{d}(\mathcal{M}_\mathcal{A}(x), x_{k+1})^2 \big| x\right] \geq \frac{1}{3}\left(\frac{1}{2\sqrt{2}} - \tau\right)^2$. Finally, we observe that it is not possible for $\mathbb{E}_x \mathbb{P}_{\mathcal{P}_r}\left[\mathcal{L}(\mathcal{M}_\mathcal{A}(x), \mathcal{P}_r) < \epsilon \big| x\right]$ to be more than $1 - \delta$ if $\inf_x \mathbb{P}_{\mathcal{P}_r}\left[\mathcal{L}(\mathcal{M}_\mathcal{A}(x), \mathcal{P}_r) \big| x\right] > \epsilon + 2\delta$, because $\mathcal{L}(\mathcal{M}_\mathcal{A}(x), \mathcal{P}_r)$ is bounded above by 2. $\qquad\square$

# 8 Algorithmic implications

## 8.1 $k-$means

Applying Theorem 4 to the case when $\mathcal{P}$ is a distribution supported equally on $n$ specific points (that are part of an input) in a unit ball of $\mathbb{R}^m$, we see that in order to obtain an additive $\epsilon$ approximation for the $k-$means problem with probability $1 - \delta$, it suffices to sample

$$s \geq C\left(\frac{k}{\epsilon^2}\left(\frac{\log^4\left(\frac{k}{\epsilon}\right)}{\epsilon^2}, k\right) + \frac{1}{\epsilon^2} \log \frac{1}{\delta}\right)$$

points uniformly at random (which would have a cost of $O(s \log n)$ if the cost of one random bit is $O(1)$) and exhaustively solve $k-$means on the resulting subset. Supposing that a dot product between two vectors $x_i$, $x_j$ can be computed using $\tilde{m}$ operations, the total cost of sampling and then exhaustively solving $k-$means on the sample is $O(\tilde{m}sk^s \log n)$. In contrast, if one asks for a multiplicative $(1 + \epsilon)$ approximation, the best running time known depends linearly on $n$ [13]. If $\mathcal{P}$ is an unknown probability distribution, the above algorithm improves upon the best results in a natural statistical framework for clustering [5].

## 8.2 Fitting piecewise linear curves

In this subsection, we illustrate the algorithmic utility of the uniform bound in Theorem 4 by obtaining an algorithm for fitting a curve of length no more than $L$, to data drawn from an unknown probability distribution $\mathcal{P}$ supported in $B$, whose sample complexity is independent of the ambient dimension. This curve, with probability $1 - \delta$, achieves a mean squared error of less than $\epsilon$ more than the optimum. The proof of its correctness and analysis of its run-time have been deferred to the journal version. The algorithm is as follows:

1. Let $k := \lceil\frac{L}{\epsilon}\rceil$ and $s \geq C\left(\frac{k}{\epsilon^2}\left(\frac{\log^4\left(\frac{k}{\epsilon}\right)}{\epsilon^2}, k\right) + \frac{1}{\epsilon^2} \log \frac{1}{\delta}\right)$. Sample points $x_1, \ldots, x_s$ i.i.d from $\mathcal{P}$ for $s =$, and set $J := \mathrm{span}(\{x_i\}_{i=1}^s)$.

2. For every permutation $\sigma$ of $[s]$, minimize the convex objective function $\sum_{i=1}^n d(x_{\sigma(i)}, y_i)^2$ over the convex set of all $s-$tuples of points $(y_1, \ldots, y_s)$ in $J$, such that $\sum_{i=1}^{s-1} \|y_{i+1} - y_i\| \leq L$.

3. If the minimum over all $(y_1, \ldots, y_s)$ (and $\sigma$) is achieved for $(z_1, \ldots, z_s)$, output the curve obtained by joining $z_i$ to $z_{i+1}$ for each $i$ by a straight line segment.

# 9 Acknowledgements

We are grateful to Stephen Boyd for several helpful conversations.

# References

[1] Noga Alon, Shai Ben-David, Nicolò Cesa-Bianchi, and David Haussler. Scale-sensitive dimensions, uniform convergence, and learnability. *J. ACM*, 44(4):615–631, 1997.

[2] Rosa I. Arriaga and Santosh Vempala. An algorithmic theory of learning: Robust concepts and random projection. In *FOCS*, pages 616–623, 1999.

[3] Peter Bartlett. The minimax distortion redundancy in empirical quantizer design. *IEEE Transactions on Information Theory*, 44:1802–1813, 1997.

[4] Mikhail Belkin and Partha Niyogi. Laplacian eigenmaps for dimensionality reduction and data representation. *Neural Comput.*, 15(6):1373–1396, 2003.

[5] Shai Ben-David. A framework for statistical clustering with constant time approximation algorithms for k-median and k-means clustering. *Mach. Learn.*, 66(2-3):243–257, 2007.

[6] Gunnar Carlsson. Topology and data. *Bulletin of the American Mathematical Society*, 46:255–308, January 2009.

[7] Sanjoy Dasgupta. Learning mixtures of gaussians. In *FOCS*, pages 634–644, 1999.

[8] David L. Donoho and Carrie Grimes. Hessian eigenmaps: Locally linear embedding techniques for high-dimensional data. *Proceedings of the National Academy of Sciences*, 100(10):5591–5596, May 2003.

[9] A. Gray. *Tubes*. Addison-Wesley, 1990.

[10] Trevor J. Hastie and Werner Stuetzle. Principal curves. *Journal of the American Statistical Association*, 84:502–516, 1989.

[11] William Johnson and Joram Lindenstrauss. Extensions of lipschitz mappings into a hilbert space. *Contemporary Mathematics*, 26:419–441, 1984.

[12] Balázs Kégl, Adam Krzyzak, Tamás Linder, and Kenneth Zeger. Learning and design of principal curves. *IEEE Transactions on Pattern Analysis and Machine Intelligence*, 22:281–297, 2000.

[13] Amit Kumar, Yogish Sabharwal, and Sandeep Sen. A simple linear time $(1+\epsilon)-$approximation algorithm for k-means clustering in any dimensions. In *FOCS*, pages 454–462, 2004.

[14] Andreas Maurer and Massimiliano Pontil. Generalization bounds for k-dimensional coding schemes in hilbert spaces. In *ALT*, pages 79–91, 2008.

[15] H. Narayanan and P. Niyogi. On the sample complexity of learning smooth cuts on a manifold. In *Proc. of the 22nd Annual Conference on Learning Theory (COLT)*, June 2009.

[16] Partha Niyogi, Stephen Smale, and Shmuel Weinberger. Finding the homology of submanifolds with high confidence from random samples. *Discrete & Computational Geometry*, 39(1-3):419–441, 2008.

[17] Sam T. Roweis and Lawrence K. Saul. Nonlinear dimensionality reduction by locally linear embedding. *SCIENCE*, 290:2323–2326, 2000.

[18] Alexander J. Smola, Sebastian Mika, Bernhard Schölkopf, and Robert C. Williamson. Regularized principal manifolds. *J. Mach. Learn. Res.*, 1:179–209, 2001.

[19] J. B. Tenenbaum, V. Silva, and J. C. Langford. A Global Geometric Framework for Nonlinear Dimensionality Reduction. *Science*, 290(5500):2319–2323, 2000.

[20] Afra Zomorodian and Gunnar Carlsson. Computing persistent homology. *Discrete & Computational Geometry*, 33(2):249–274, 2005.

